# Value-Directed Compression of POMDPs

**Pascal Poupart**
Departement of Computer Science
University of Toronto
Toronto, ON, M5S 3H5
*ppoupart@cs.toronto.edu*

**Craig Boutilier**
Department of Computer Science
University of Toronto
Toronto, ON, M5S 3H5
*cebly@cs.toronto.edu*

## Abstract

We examine the problem of generating state-space compressions of POMDPs in a way that minimally impacts decision quality. We analyze the impact of compressions on decision quality, observing that compressions that allow accurate policy evaluation (prediction of expected future reward) will not affect decision quality. We derive a set of sufficient conditions that ensure accurate prediction in this respect, illustrate interesting mathematical properties these confer on lossless linear compressions, and use these to derive an iterative procedure for finding good linear lossy compressions. We also elaborate on how structured representations of a POMDP can be used to find such compressions.

## 1 Introduction

Partially observable Markov decision processes (POMDPs) provide a rich framework for modeling a wide range of sequential decision problems in the presence of uncertainty. Unfortunately, the application of POMDPs to real world problems remains limited due to the intractability of current solution algorithms, in large part because of the exponential growth of state spaces with the number of relevant variables.

Ideally, we would like to mitigate this source of intractability by compressing the state space as much as possible without compromising decision quality. Our aim in solving a POMDP is to maximize future reward based on our current beliefs about the world. By compressing its *belief state*, an agent may lose relevant information, which results in suboptimal policy choice. Thus an important aspect of belief state compression lies in distinguishing relevant information from that which can be safely discarded. A number of schemes have been proposed for either directly or indirectly compressing POMDPs. For example, approaches using bounded memory [8, 10] and state aggregation—either dynamic [2] or static [5, 9]—can be viewed in this light.

In this paper, we study the effect of static state-space compression on decision quality. We first characterize *lossless* compressions—those that do not lead to any error in expected value—by deriving a set of conditions that guarantee decision quality will not be impaired. We also characterize the specific case of linear compressions. This analysis leads to algorithms that find good compression schemes, including methods that exploit structure in the POMDP dynamics (as exhibited, e.g., in graphical models). We then extend these concepts to *lossy* compressions. We derive a (somewhat loose) upper bound on the loss in decision quality when the conditions for lossless compression (of some required dimensionality) are

not met. Finally we propose a simple optimization program to find linear lossy compressions that minimizes this bound, and describe how structured POMDP models can be used to implement this scheme efficiently.

## 2 Background and Notation

### 2.1 POMDPs

A POMDP is defined by: a set $\mathcal{S}$ of states $s$; a set $\mathcal{A}$ of actions $a$; a set $\mathcal{Z}$ of observations $z$; a transition function $T$, where $T(s, a, s')$ denotes the transition probability $Pr(s'|s, a)$; an observation function $Z$, where $Z(s, z)$ denotes the probability $Pr(z|s)$ of making observation $z$ in state $s$; and a reward function $R$, where $R(s)$ denotes the immediate reward associated with state $s$.[1] We assume discrete state, action and observation sets and we focus on discounted, infinite horizon POMDPs with discount factor $0 \leq \gamma < 1$.

Policies and value functions for POMDPs are typically defined over *belief space*, where a belief state $b$ is a distribution over $\mathcal{S}$ capturing an agent's knowledge about the current state of the world. Belief state $b$ can be updated in response to a specific action-observation pair $\langle a, z \rangle$ using Bayes rule: $b'(s') = \alpha \sum_s b(s)T(s, a, s')Z(s', z)$ ($\alpha$ is a normalization constant). We denote the (unnormalized) mapping $T^{a,z}$, where, in matrix form, we have $T_{ij}^{a,z} = Pr(s_j|a, s_i)Pr(z|s_j)$. Note that a belief state $b$ and reward function $R$ can be viewed respectively as $|\mathcal{S}|$-dimensional row and column vectors. We define $R(b) = b \cdot R$.

Solving a POMDP consists of finding an optimal policy $\pi$ mapping belief states to actions. The value $V^\pi$ of a policy $\pi$ is the expected sum of discounted rewards and is defined as:

$$V^\pi(b) = R(b) + \gamma \sum_z V^\pi(T^{\pi(b),z}(b)) \tag{1}$$

A number of techniques [11] based on value iteration or policy iteration can be used to compute optimal or approximately optimal policies for POMDPs.

### 2.2 Conditional Independence and Additive Separability

When our state space is defined by a set of variables, POMDPs can often be represented concisely in a factored way by specifying the transition, observation and reward functions using a *dynamic Bayesian network (DBN)*. Such representations exploit the fact that transitions associated with each variable depend only on a small subset of variables. These representations can often be exploited to solve POMDPs without state space enumeration [2].

Recently, Pfeffer [13] showed that conditional independence combined with some form of additive separability can enable efficient inference in many DBNs. Roughly, a function can be *additively separated* when it decomposes into a sum of smaller terms. For instance, $Pr(Z|XY)$ is separable if there exist conditional distributions $Pr_X(Z|X)$ and $Pr_Y(Z|Y)$, and $\alpha \in [0, 1]$, such that $Pr(Z|XY) = \alpha Pr_X(Z|X) + (1 - \alpha)Pr_Y(Z|Y)$. This ensures that one need only know the marginals of $X$ and $Y$ (instead of their joint distribution) to infer $Z$. Pfeffer shows how additive separability in the CPTs of a DBN can be exploited to identify families of *self-sufficient variables*. A self-sufficient family consists of a set of subsets of variables such that the marginals of each subset are sufficient to predict the marginals of the same subsets at the next time step. Hence, if we require the probabilities of a few variables, and can identify a self-sufficient family containing those variables, then we need only compute marginals over this family when monitoring belief state.

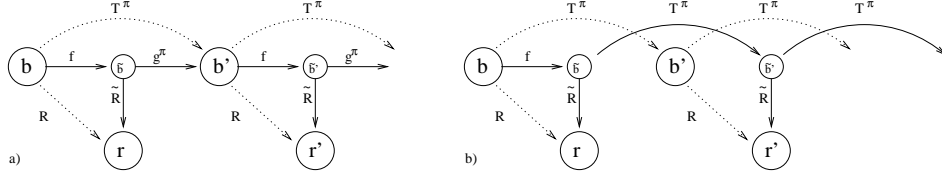

Figure 1: a) Functional flow of a POMDP (dotted arrows) and a compressed POMDP (solid arrows) where the next belief state is accurately predicted. b) Functional flow of a POMDP (dotted arrows) and a compressed POMDP (solid arrows) where the next compressed belief state is accurately predicted.

### 2.3 Invariant and Krylov Subspaces

We briefly review several linear algebraic concepts used later (see [15] for more details). Let $S$ be a vector subspace. We say $S$ is *invariant* with respect to matrix $M$ if it is closed under multiplication by $M$ (i.e., $Mx \in S, \forall x \in S$). A *Krylov subspace* $Kr(M, x)$ is the smallest subspace $S$ that contains $x$ and is invariant with respect to $M$. A basis $B$ for a Krylov subspace can easily be generated by repeatedly multiplying $x$ by $M$ (i.e., $B = \{x, Mx, M^2x, M^3x, \ldots\}$). If $Kr(M, x)$ is $n$-dimensional, one can show that $M^{n-1}x$ is the last linearly independent vector in this sequence and that all subsequent vectors are linear combinations of $B$.

In a DBN, families of self-sufficient variables naturally correspond to invariant subspaces. For instance, suppose $f$ is a linear function that depends only on self-sufficient family $\{\{X\}, \{Y, Z\}\}$. If we regress $f$ through the dynamics of the DBN—i.e., if we multiply $f$ by the transition matrix $T^{a,z}$—the resulting function will also be defined over the truth values of $\{X\}$ and $\{Y, Z\}$. Hence, when a family of variables is self-sufficient, the subspace of linear functions defined over the truth values of that family is invariant w.r.t. $T^{a,z}$.

## 3 Lossless Compressions

If a compression of the state space of a POMDP allows us to accurately evaluate all policies, we say the compression is *lossless*, since we have sufficient information to select the optimal policy. We provide one characterization of lossless compressions. We then specialize this to the linear case, and discuss the use of compact POMDP representations.

Let $f$ be a *compression function* that maps each belief state $b$ into some lower dimensional compressed belief state $\tilde{b}$ (see Figure 1(a)). Here $\tilde{b}$ can be viewed as a *bottleneck* (e.g., in the sense of the information bottleneck [17]) that filters the information contained in $b$ before it's used to estimate future rewards. We desire a compression $f$ such that $\tilde{b}$ corresponds to the smallest statistic sufficient for accurately predicting the current reward $r$ as well as the next belief state $b'$ (since we can accurately predict all following rewards from $b'$). Such a compression $f$ exists if we can also find mappings $g^{a,z}$ and $\tilde{R}$ such that:

$$R = \tilde{R} \circ f \quad \text{and} \quad T^{a,z} = g^{a,z} \circ f \ \ \forall a \in \mathcal{A}, z \in \mathcal{Z} \tag{2}$$

Since we are only interested in predicting future rewards, we don't really need to accurately estimate the next belief state $b'$; we could just predict the next compressed belief state $\tilde{b}'$ since it captures all information in $b'$ relevant for estimating future rewards. Figure 1(b) illustrates the resulting functional flow, where $\tilde{T}^{a,z}$ represents the transition function that directly maps one compressed belief state to the next compressed belief state. Eq. 2 can

then be replaced by the following weaker but still sufficient conditions:

$$R = \tilde{R} \circ f \quad \text{and} \quad f \circ T^{a,z} = \tilde{T}^{a,z} \circ f \quad \forall a \in \mathcal{A}, z \in \mathcal{Z} \tag{3}$$

Given an $f$, $\tilde{R}$ and $\tilde{T}^{a,z}$ satisfying Eq. 3, we can evaluate a policy $\pi$ using the compressed POMDP dynamics as follows:

$$\tilde{V}^{\pi}(\tilde{b}) = \tilde{R}(\tilde{b}) + \gamma \sum_z \tilde{V}^{\pi}(\tilde{T}^{\pi(\tilde{b}),z}(\tilde{b})) \tag{4}$$

Once $\tilde{V}^{\pi}$ is found, we can recover the original value function $V^{\pi} = \tilde{V}^{\pi} \circ f$. Indeed, Eq. 1 and Eq. 4 are equivalent:

**Theorem 1** *Let $f$, $\tilde{R}$ and $\tilde{T}^{a,z}$ satisfy Eq. 3 and let $V^{\pi} = \tilde{V}^{\pi} \circ f$. Then Eq. 1 holds iff Eq. 4 does.*

**Proof**

$$
\begin{aligned}
& V^{\pi}(b) = R^{\pi}(b) + \gamma \sum_z V^{\pi}(T^{\pi(b),z}(b)) \\
\iff & \tilde{V}^{\pi}(f(b)) = \tilde{R}(f(b)) + \gamma \sum_z \tilde{V}^{\pi}(f(T^{\pi(b),z}(b))) \\
\iff & \tilde{V}^{\pi}(f(b)) = \tilde{R}(f(b)) + \gamma \sum_z \tilde{V}^{\pi}(\tilde{T}^{\pi(b),z}(f(b))) \\
\iff & \tilde{V}^{\pi}(\tilde{b}) = \tilde{R}(\tilde{b}) + \gamma \sum_z \tilde{V}^{\pi}(\tilde{T}^{\pi(b),z}(\tilde{b}))
\end{aligned}
$$

## 3.1 Linear compressions

We say $f$ is a *linear compression* when $f$ is a linear function, representable by some matrix $F$. In this case, the approximate transition and reward functions $\tilde{T}^{a,z}$ and $\tilde{R}$ must also be linear (assuming Eq. 3 is satisfied). Eq. 3 can be rewritten in matrix notation:

$$R = F\tilde{R} \quad \text{and} \quad T^{a,z}F = F\tilde{T}^{a,z} \quad \forall a, z \tag{5}$$

In a linear compression, $F$ can be viewed as effecting a change of basis for the value function, with the columns of $F$ defining a subspace in which the compressed value function lies. Furthermore, the rank of $F$ indicates the dimensionality of the compressed state space. When Eq. 5 is satisfied, the columns of $F$ span a subspace that contains $R$ and that is invariant with respect to each $T^{a,z}$. Intuitively, Eq. 5 says that a sufficient statistic must be able to "predict itself" at the next time step (hence the subspace is invariant), and that it must predict the current reward (hence the subspace contains $R$). Formally:

**Theorem 2** *Let $T^{\tilde{a},z}$, $\tilde{R}$ and $F$ satisfy Eq. 5. Then the range of $F$ contains $R$ and is invariant with respect to each $T^{a,z}$.*

**Proof** Eq. 5 ensures $R$ is a linear combination of the columns of $F$, so it lies in the range of $F$. It also requires that the columns of each $T^{a,z}F$ are linear combinations of the columns of $F$, so $F$ is invariant with respect to each $T^{a,z}$.

Thus, the best linear lossless compression corresponds to the smallest invariant subspace that contains $R$. This is by definition the Krylov subspace $Kr(\{T^{a,z} : a \in \mathcal{A}, z \in \mathcal{Z}\}, R)$. Using this fact we can easily compute the best lossless linear compression by iteratively multiplying $R$ by each $T^{a,z}$ until the Krylov basis is obtained. We then let the Krylov basis form the columns of $F$, and compute $\tilde{R}$ and each $\tilde{T}^{a,z}$ by solving each part of Eq. 5. Finally, we can solve the POMDP in the compressed state space by using $\tilde{R}$ and $\tilde{T}^{a,z}$.

Note that this technique can be viewed as a generalization of Givan et al's MDP model minimization technique [3]. It is interesting to note that Littman et al. [9] proposed a similar iterative algorithm to compress POMDPs based on predicting future observations.[2]

## 3.2 Structured Linear Compressions

When a POMDP is specified in compactly, say, using a DBN, the size of the state space may be exponentially larger than the specification. The practical need to avoid state enumeration is a key motivation for POMDP compression. However, the complexity of the search for a good compression must also be independent of the state space size. Unfortunately, the iterative Krylov algorithm involves repeatedly multiplying explicit transition matrices and basis vectors. We consider several ways in which a compact POMDP specification can be exploited to construct a linear compression without state enumeration.

One solution lies in exploiting DBN structure and context-specific independence. If transition, observation and reward functions are represented using DBNs and structured CPTs (e.g., decision trees or algebraic decision diagrams), then the matrix operations required by the Krylov algorithm can be implemented effectively [1, 7]. Although this approach can offer substantial savings, the DTs or ADDs that represent the basis vectors of the Krylov subspace may still be much larger than the dimensionality of the compressed state space and the original DBN specifications.

Alternatively, families of self-sufficient variables corresponding to invariant subspaces can be identified by exploiting additive separability. Starting with the variables upon which $R$ depends, we can recursively grow a family of variables until it is self-sufficient with respect to each $T^{a,z}$. The corresponding subspace is invariant and necessarily contains $R$. Assuming a tractable self-sufficient family is found, a compact basis can then be constructed by using all *indicator functions* for each subset of variables in this family (e.g., if $\{X, Y, Z\}$ is one such subset of binary variables, then eight basis vectors will correspond to this set). This approach allows us to quickly identify a good compression by a simple inspection of the additive separability structure of the DBN. The resulting compression is not necessarily optimal; however, it is the best among those corresponding to some such family. It is important to note that the dynamics $\tilde{T}^{a,z}$ and reward $\tilde{R}$ of the compressed POMDP can be constructed easily (i.e., without state enumeration) from this $F$ and the original DBN model. Pfeffer [13] notes that observations tend to reduce the amount of additive separability present in a DBN, thereby increasing the size of self-sufficient families. Therefore, we should point out that lossless compressions of POMDPs that exploit self-sufficiency and offer an acceptable degree of compression may not exist. Hence lossy compressions are likely to be required in many cases.

Finally, we ask whether the existence of lossless compressions requires some form of structure in the POMDP. We argue that this is almost always the case. Suppose a transition matrix $T^{a,z}$ and a reward vector $R$ are chosen uniformly at random. The odds that $R$ falls into a proper invariant subspace of $T^{a,z}$ are essentially zero since there are infinitely more vectors in the full space than in all the proper invariant subspaces put together. This means that if a POMDP can be compressed, it must almost certainly be because its dynamics exhibit some structure. We have described how context-specific independence and additive separability can be exploited to identify some linear lossless compressions. However they do not guarantee that the optimal compression will be found, so it remains an open question whether other types of structure could be used in similar ways.

## 4 Lossy compressions

Since we cannot generally find effective lossless compressions, we also consider lossy compressions. We propose a simple approach to find linear lossy compressions that "almost satisfy" Eq. 5. Table 1 outlines a simple optimization program to find lossy compressions that minimize a weighted sum of the max-norm residual errors, $\epsilon_T$ and $\epsilon_R$, in Eq. 5. Here $c$ and $d$ are weights that allow us to vary the degree to which the two components of Eq. 5

$$\min \quad c\epsilon_R + d\epsilon_T$$
$$\text{s.t.} \quad -\epsilon_R \leq \|R - F\tilde{R}\|_\infty \leq \epsilon_R \tag{6}$$
$$-\epsilon_T \leq \|T^{a,z}F - F\tilde{T}^{a,z}\|_\infty \leq \epsilon_T \quad \forall a \in \mathcal{A}, z \in \mathcal{Z} \tag{7}$$
$$\|F\|_\infty = 1$$

Table 1: Optimization program for linear lossy compressions

should be satisfied. The unknowns of the program are all the entries of $\tilde{R}$, $\tilde{T}^{a,z}$ and $F$ as well as $\epsilon_T$ and $\epsilon_R$. The constraint $\|F\|_\infty = 1$ is necessary to preserve scale, otherwise $\epsilon_T$ could be driven down to 0 simply by setting all the entries of $F$ to 0. Since $\tilde{T}^{a,z}$ and $\tilde{R}$ multiply $F$, some constraints are nonlinear. However, it is possible to solve this optimization program by solving a series of LPs (linear programs). We alternate solving the LP that adjusts $\tilde{R}$ and $\tilde{T}^{a,z}$ while keeping $F$ fixed, and solving the LP that adjusts $F$ while keeping $\tilde{R}$ and $\tilde{T}^{a,z}$ fixed. This guarantees that the objective function decreases at each iteration and will converge, but not necessarily to a local optimum.

## 4.1 Max-norm Error Bound

The quality of the compression resulting from this program depends on the weights $c$ and $d$. Ideally, we would like to set $c$ and $d$ in a way that $c\epsilon_R + d\epsilon_T$ represents the loss in decision quality associated with compressing the state space. If we can bound the error $\epsilon_V$ of evaluating any policy using the compressed POMDP, then the difference in expected total return between the policy that is best w.r.t. the compressed POMDP and the true optimal policy is at most $2\epsilon_V$. Let $\epsilon_V$ be $\max_\pi \|V^\pi - \tilde{V}^\pi \circ f\|_\infty$. Theorem 3 gives an upper bound on $\epsilon_V$ as a linear combination of the max-norm residual errors in Eq. 5.

**Theorem 3** *Let* $\epsilon_V = \max_\pi \|V^\pi - \tilde{V}^\pi \circ f\|_\infty$, $\epsilon_R = \|R - \tilde{R} \circ f\|_\infty$, $\epsilon_T = \max_{a,z} \|T^{a,z} - \tilde{T}^{a,z} \circ f\|_\infty$ *and* $\tilde{V}^* = max_\pi \tilde{V}^\pi$. *Then* $\epsilon_V \leq \frac{1}{1-\gamma}\epsilon_R + \frac{\gamma|\mathcal{Z}| \|\tilde{V}^*\|_\infty}{1-\gamma}\epsilon_T$.

We omit the proof due to lack of space. It essentially consists of a sequence of substitutions of the type $\|AB\|_\infty \leq \|A\|_\infty \|B\|_\infty$ and $\|A + B\|_\infty \leq \|A\|_\infty + \|B\|_\infty$. We suspect that the above error bound will grossly overestimate the loss in decision quality, however we intend to use it mostly as a guide for setting $c$ and $d$. Here $\gamma|\mathcal{Z}| \|\tilde{V}^*\|_\infty/(1-\gamma)$ is typically much greater than $1/(1-\gamma)$ because of the factor $\|\tilde{V}^*\|_\infty$, which means that $\epsilon_T$ has a much higher impact on the loss in decision quality than $\epsilon_R$. Intuitively, this makes sense because the error $\epsilon_T$ in predicting the next compressed belief state may compound over time, so we should set $d$ significantly higher than $c$.

## 4.2 Structured Compressions

As with lossless compressions, solving the program in Table 1 may be intractable due to the size of $\mathcal{S}$. There are $O(|\mathcal{S}|)$ constraints and $|\mathcal{S}||\tilde{\mathcal{S}}|$ unknown entries in matrix $F$.[3] We describe several techniques that allow one to exploit problem structure to find an acceptable lossy compression without state space enumeration.

One approach is related to the basis function model proposed in [4], in which we restrict $F$ to be functions over some small set of *factors* (subsets of state variables.) This ensures that the number of unknown parameters in any column of $F$ (which we optimize in Table 1) is

linear in the number of instantiations of each factor. By keeping factors small, we maintain a manageable set of unknowns. To deal with the $O(|\mathcal{S}|)$ constraints, we can exploit the structure imposed on $F$ and the DBN structure to reduce the number of constraints to something (in the many cases) polynomial in the number of state variables. This can be achieved using the techniques described in [4, 16] to rewrite an LP with many fewer constraints or to generate small subsets of constraints incrementally. These techniques are rather involved, so we refer to the cited papers for details.

By searching within a restricted set of structured compressions and by exploiting DBN structure it is possible to efficiently solve the optimization program in Table 1. The question of factor selection remains: on what factors should $F$ be defined? A version of this question has been tackled in [12, 14] in the context of selecting a basis to approximately solve MDPs. The techniques proposed in those papers could be adapted to our optimization program.

An alternative method for structuring the computation of $F$ involves additive separability. Let $\mathbf{X}_j$ ($j \leq m$) be subsets of variables, and $\phi_j(\mathbf{X}_j, \tilde{S})$ be a function over $\mathbf{X}_j$ and the compressed state space $\tilde{S}$. We restrict each column of $F$ to be a separable function of the $\phi_j$; that is, column $i$ (corresponding to state $\tilde{s}_i$) is $\sum_j \beta_j \phi_j(\mathbf{X}_j, \tilde{s}_i)$ for some parameters $\beta_j$. Here the $\beta_j$ can be viewed as weights indicating the importance of the contribution of each $\phi_j$ in the separable function. Given a family of subsets, the parameters over which we optimize to determine $F$ are now the $\beta_j$ and the entries of each function $\phi_j(\mathbf{X}_j, \tilde{S})$. While nonlinear, the same alternating minimization scheme described earlier can be used to optimize these two classes of parameters of $F$ in turn. Note that the number of variables is dependent only on the size of the subsets $\mathbf{X}_j$ and the compressed state space $\tilde{S}$. Furthermore, this form of additive separability lends itself to the same compact constraint generation techniques mentioned above. Finally, the (discrete) search for decent subsets $\mathbf{X}_j$ can be interleaved with optimization of the compression mapping for fixed sets $\mathbf{X}_j$.

## 5 Preliminary Experiments

We report on preliminary experiments with the coffee problem described in [2]. Given its relatively small size (32 states, 3 observations and 2 actions), these results should be viewed as simply illustrating the feasibility and potential of the algorithms proposed in Secs. 3.1 and 4.1. Further experiments for the structured versions (Secs. 3.2 and 4.2) are necessary to assess the degree of compression achievable with large, realistic problems.

The 32-dimensional belief space can be compressed without any loss to a 7-dimensional subspace using the Krylov subspace algorithm described in Section 3.1. For further compression, we applied the optimization program described in Table 1 by setting the weights $c$ and $d$ to 1 and 200 respectively. The alternating variable technique was iterated 150 times, with the best solution chosen from 15 random restarts (to mitigate the effects of local optima). Figure 2 shows the loss in expected return (w.r.t. the optimal policy) when policy computed using varying degrees of compression is executing for 100 stages. The loss is sampled from 100,000 random initial belief states, averaged over 10 runs. These policies manage to achieve expected returns with less than 4% loss. In contrast, the average loss of a random policy is about 2.5 (or 27%).

## 6 Concluding Remarks

We have presented an in-depth theoretical analysis of the impact of static compressions on decision quality. We derived a set of conditions that guarantee compression does not impair decision quality, leading to interesting mathematical properties for linear compressions that allow us to exploit structure in the POMDP dynamics. We also proposed a simple

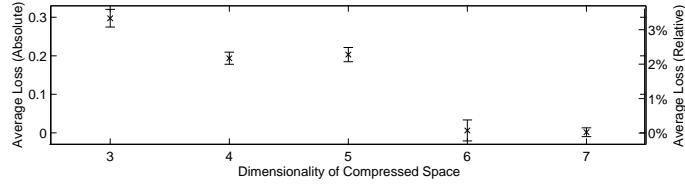

Figure 2: Average loss for various lossy compressions

optimization program to search for good lossy compressions. Preliminary results suggest that significant compression can be achieved with little impact on decision quality.

This research can be extended in various directions. It would be interesting to carry out a similar analysis in terms of information theory (instead of linear algebra) since the problem of identifying information in a belief state relevant to predicting future rewards can be modeled naturally using information theoretic concepts [6]. Dynamic compressions could also be analyzed since, as we solve a POMDP, the set of reasonable policies shrinks, allowing greater compression.

## Footnotes

[1]The ideas presented in this paper generalize to cases when $Z$ and $R$ also depend on actions.

[2]Assuming that rewards are functions of the observations.

[3]Assuming $\tilde{\mathcal{S}}$ is small, the $|\tilde{\mathcal{S}}|^2$ variables in each $\tilde{T}^{a,z}$ and $|\tilde{\mathcal{S}}|$ variables in $\tilde{R}$ are unproblematic.

# References

[1] C. Boutilier, R. Dearden, and M. Goldszmidt. Stochastic dynamic programming with factored representations. *Artificial Intelligence*, 121:49–107, 2000.

[2] C. Boutilier and D. Poole. Computing optimal policies for partially observable decision processes using compact representations. *Proc. AAAI-96*, pp.1168–1175, Portland, OR, 1996.

[3] R. Givan, T. Dean, and M. Greig. Equivalence notions and model minimization in Markov decision processes. *Artificial Intelligence*, to appear, 2002.

[4] C. Guestrin, D. Koller, and R. Parr. Max-norm projections for factored MDPs. *Proc. IJCAI-01*, pp.673–680, Seattle, WA, 2001.

[5] C. Guestrin, D. Koller, and R. Parr. Solving factored POMDPs with linear value functions. *IJCAI-01 Worksh. on Planning under Uncertainty and Inc. Info.*, Seattle, WA, 2001.

[6] C. Guestrin and D. Ormoneit. Information-theoretic features for reinforcement learning. Unpublished manuscript.

[7] J. Hoey, R. St-Aubin, A. Hu, and C. Boutilier. SPUDD: Stochastic planning using decision diagrams. *Proc. UAI-99*, pp.279–288, Stockholm, 1999.

[8] M. L. Littman. Memoryless policies: theoretical limitations and practical results. In D. Cliff, P. Husbands, J. Meyer, S. W. Wilson, eds., *Proc. 3rd Intl. Conf. Sim. of Adaptive Behavior*, Cambridge, 1994. MIT Press.

[9] M. L. Littman, R. S. Sutton, and S. Singh. Predictive representations of state. *Proc.NIPS-02*, Vancouver, 2001.

[10] R. A. McCallum. Hidden state and reinforcement learning with instance-based state identification. *IEEE Transactions on Systems, Man, and Cybernetics*, 26(3):464–473, 1996.

[11] K. Murphy. A survey of POMDP solution techniques. Technical Report, U.C. Berkeley, 2000.

[12] R. Patrascu, P. Poupart, D. Schuurmans, C. Boutilier, C. Guestrin. Greedy linear value-approximation for factored Markov decision processes. *AAAI-02*, pp.285–291, Edmonton, 2002.

[13] A. Pfeffer. Sufficiency, separability and temporal probabilistic models. *Proc. UAI-01*, pp.421–428, Seattle, WA, 2001.

[14] P. Poupart, C. Boutilier, R. Patrascu, and D. Schuurmans. Piecewise linear value function approximation for factored MDPs. *AAAI-02*, pp.292–299, Edmonton, 2002.

[15] Y. Saad. *Iterative Methods for Sparse Linear Systems*. PWS, Boston, 1996.

[16] D. Schuurmans and R. Patrascu. Direct value-approximation for factored MDPs. *Proc. NIPS-01*, Vancouver, 2001.

[17] N. Tishby, F. C. Pereira, and W. Bialek. The information bottleneck method. *37th Annual Allerton Conf. on Comm., Contr. and Computing*, pp.368–377, 1999.
